# Receptive field formation in natural scene environments: comparison of single cell learning rules

**Brian S. Blais**
Brown University Physics Department
Providence, RI 02912

**N. Intrator**
School of Mathematical Sciences
Tel-Aviv University
Ramat-Aviv, 69978 ISRAEL

**H. Shouval**
Institute for Brain and Neural Systems
Brown University
Providence, RI 02912

**Leon N Cooper**
Brown University Physics Department and
Institute for Brain and Neural Systems
Brown University
Providence, RI 02912

## Abstract

We study several statistically and biologically motivated learning rules using the same visual environment, one made up of natural scenes, and the same single cell neuronal architecture. This allows us to concentrate on the feature extraction and neuronal coding properties of these rules. Included in these rules are kurtosis and skewness maximization, the quadratic form of the BCM learning rule, and single cell ICA. Using a structure removal method, we demonstrate that receptive fields developed using these rules depend on a small portion of the distribution. We find that the quadratic form of the BCM rule behaves in a manner similar to a kurtosis maximization rule when the distribution contains kurtotic directions, although the BCM modification equations are computationally simpler.

# 1    INTRODUCTION

Recently several learning rules that develop simple cell-like receptive fields in a natural image environment have been proposed (Law and Cooper, 1994; Olshausen and Field, 1996; Bell and Sejnowski, 1997). The details of these rules are different as well as their computational reasoning, however they all depend on statistics of order higher than two and they all produce sparse distributions.

In what follows we investigate several specific modification functions that have the. general properties of BCM synaptic modification functions (Bienenstock et al., 1982), and study their feature extraction properties in a natural scene environment. Several of the rules we consider are derived from standard statistical measures (Kendall and Stuart, 1977), such as skewness and kurtosis, based on polynomial moments. We compare these with the quadratic form of BCM (Intrator and Cooper, 1992), though one should note that this is not the only form that could be used. By subjecting all of the learning rules to the same input statistics and retina/LGN preprocessing and by studying in detail the single neuron case, we eliminate possible network/lateral interaction effects and can examine the properties of the learning rules themselves.

We compare the learning rules and the receptive fields they form, and introduce a procedure for directly measuring the sparsity of the representation a neuron learns. This gives us another way to compare the learning rules, and a more quantitative measure of the concept of sparse representations.

# 2    MOTIVATION

We use two methods for motivating the use of the particular rules.  One comes from Projection Pursuit (Friedman, 1987) and the other is Independent Component Analysis (Comon, 1994).  These methods are related, as we shall see, but they provide two different approaches for the current work.

## 2.1   EXPLORATORY PROJECTION PURSUIT

Diaconis and Freedman (1984) show that for most high-dimensional clouds (of points), most low-dimensional projections are approximately Gaussian. This finding suggests that important information in the data is conveyed in those directions whose single dimensional projected distribution is far from Gaussian.

Intrator (1990) has shown that a BCM neuron can find structure in the input distribution that exhibits deviation from Gaussian distribution in the form of multi-modality in the projected distributions. This type of deviation is particularly useful for finding clusters in high dimensional data.   In the natural scene environment, however, the structure does not seem to be contained in clusters. In this work we show that the BCM neuron can still find interesting structure in non-clustered data.

The most common measures for deviation from Gaussian distribution are skewness and. kurtosis which are functions of the first three and four moments of the distribution respectively. Rules based on these statistical measures satisfy the BCM conditions proposed in Bienenstock et al. (1982), including a threshold-based stabilization. The details of these rules and some of the qualitative features of the stabilization are different, however.   In addition, there are some learning rules, such as the ICA rule of Bell and Sejnowski (1997) and the sparse coding algorithm of Olshausen and Field (1995), which have been used with natural scene inputs to produce oriented receptive fields. We do not include these in our comparison be-

cause they are not single cell learning rules, and thus detract from our immediate goal of comparing rules with the same input structure and neuronal architecture.

## 2.2 INDEPENDENT COMPONENT ANALYSIS

Recently it has been claimed that the independent components of natural scenes are the edges found in simple cells (Bell and Sejnowski, 1997). This was achieved through the maximization of the mutual entropy of a set of mixed signals. Others (Hyvarinen and Oja, 1996) have claimed that maximizing kurtosis can also lead to the separation of mixed signals into independent components. This alternate connection between kurtosis and receptive fields leads us into a discussion of ICA.

Independent Component Analysis (ICA) is a statistical signal processing technique whose goal is to express a set of random variables as a linear mixture of statistically independent variables. The problem of ICA is then to find the transformation from the observed mixed signals to the "unmixed" independent sources. The search for independent components relies on the fact that a linear mixture of two non-Gaussian distributions will become more Gaussian than either of them. Thus, by seeking projections which maximize deviations from Gaussian distribution, we recover the original (independent) signals. This explains the connection of ICA to the framework of exploratory projection pursuit.

## 3  SYNAPTIC MODIFICATION RULES

In this section we outline the derivation for the learning rules in this study. Neural activity is assumed to be a positive quantity, so for biological plausibility we denote by $c$ the rectified activity $\sigma(\mathbf{d} \cdot \mathbf{m})$, where $\sigma(\cdot)$ is a smooth monotonic function with a positive output (a slight negative output is also allowed). $\sigma'$ denotes the derivative of the sigmoidal. The rectification is required for all rules that depend on odd moments because these vanish in symmetric distributions such as natural scenes. We study the following measures(Kendall and Stuart, 1977, for review):

**Skewness 1**  This measures the deviation from symmetry, and is of the form:
$$S_1 = E[c^3]/E^{1.5}[c^2]. \tag{1}$$
A maximization of this measure via gradient ascent gives
$$\nabla S_1 = \frac{1}{\Theta_M^{1.5}} E\left[c\left(c - E[c^3]/E[c^2]\right)\sigma'\mathbf{d}\right] = \frac{1}{\Theta_M^{1.5}} E\left[c\left(c - E[c^3]/\Theta_M\right)\sigma'\mathbf{d}\right] \tag{2}$$
where $\Theta_m$ is defined as $E[c^2]$.

**Skewness 2**  Another skewness measure is given by
$$S_2 = E[c^3] - E^{1.5}[c^2]. \tag{3}$$
This measure requires a stabilization mechanism which we achieve by requiring that the vector of weights, denoted by $m$, has norm of 1. The gradient of $S_2$ is
$$\nabla S_2 = 3E\left[c^2 - c\sqrt{E[c^2]}\right] = 3E\left[c\left(c - \sqrt{\Theta_M}\right)\sigma'\mathbf{d}\right], \parallel \mathbf{m} \parallel = 1 \tag{4}$$

**Kurtosis 1**  Kurtosis measures deviation from Gaussian distribution mainly in the tails of the distribution. It has the form
$$K_1 = E[c^4]/E^2[c^2] - 3. \tag{5}$$
This measure has a gradient of the form
$$\nabla K_1 = \frac{1}{\Theta_M^2} E\left[c\left(c^2 - E[c^4]/E[c^2]\right)\sigma'\mathbf{d}\right] = \frac{1}{\Theta_M^2} E\left[c\left(c^2 - E[c^4]/\Theta_M\right)\sigma'\mathbf{d}\right]. \tag{6}$$

**Kurtosis 2**   As before, there is a similar form which requires some stabilization:

$$K_2 = E[c^4] - 3E^2[c^2]. \tag{7}$$

This measure has a gradient of the form

$$\nabla K_2 = 4E\left[c^3 - cE[c^2]\right] = 3E\left[c(c^2 - \Theta_M)]\sigma'\mathbf{d}\right], \quad \parallel \mathbf{m} \parallel = 1. \tag{8}$$

**Kurtosis 2 and ICA**   It has been shown that kurtosis, defined as

$$K_2 = E\left[\mathbf{c}^4\right] - 3E^2\left[\mathbf{c}^2\right]$$

can be used for ICA(Hyvarinen and Oja, 1996).   Thus, finding the extrema of kurtosis of the projections enables the estimation of the independent components. They obtain the following expression

$$\mathbf{m} = \frac{2}{\lambda}\left(E^{-1}\left[\mathbf{dd}^T\right]E\left[\mathbf{d}(\mathbf{m}\cdot\mathbf{d})^3\right] - 3\mathbf{m}\right). \tag{9}$$

which  leads to an iterative "fixed-point algorithm".

**Quadratic BCM**   The Quadratic BCM (QBCM) measure as given in (Intrator and Cooper, 1992) is of the form

$$\text{QBCM} = \frac{1}{3}E[c^3] - \frac{1}{4}E^2[c^2]. \tag{10}$$

Maximizing this form using gradient ascent gives the learning rule:

$$\nabla\text{QBCM} = E\left[c^2 - cE[c^2]\right] = E[c(c - \Theta_M)\sigma'\mathbf{d}]. \tag{11}$$

# 4   METHODS

We use 13x13 circular patches from 12 images of natural scenes, presented to the neuron each iteration of the learning. The natural scenes are preprocessed either with a Difference of Gaussians (DOG) filter(Law and Cooper, 1994) or a whitening filter(Oja, 1995; Bell and Sejnowski, 1995), which eliminates the second order correlations. The moments of the output, $c$, are calculated iteratively, and when it is needed  (i.e. $K_2$ and $S_2$) we also normalize the weights at each iteration.

For Oja's fixed-point algorithm, the learning was done in batches of 1000 patterns over which the expectation values were performed. However, the covariance matrix was calculated over the entire set of input patterns.

# 5   RESULTS

## 5.1   RECEPTIVE FIELDS

The resulting receptive fields (RFs) formed are shown in Figure 1 for both the DOGed and whitened images.   Every learning rule developed oriented receptive fields, though some were more sensitive to the preprocessing than others. The additive versions of kurtosis and skewness, $K_2$ and $S_2$ respectively, developed RFs with a higher spatial frequency, and more orientations, in the whitened environment than in the DOGed environment.

The multiplicative versions of kurtosis and skewness, $K_1$ and $S_1$ respectively, as well as QBCM, sampled from many orientations regardless of the preprocessing. $S_1$ gives receptive fields with lower spatial frequencies than either QBCM or $K_1$.

This disappears with the whitened inputs, which implies that the spatial frequency of the RF is related to the dependence of the learning rule on the second moment. Example receptive fields using Oja's fixed-point ICA algorithm not surprisingly look qualitatively similar to those found using the stochastic maximization of $K_2$.

The output distributions for all of the rules appear to be double exponential. This distribution is one which we would consider sparse, but it would be difficult to compare the sparseness of the distributions merely on the appearance of the output distribution alone. In order to determine the sparseness of the code, we introduce a method for measuring it directly.

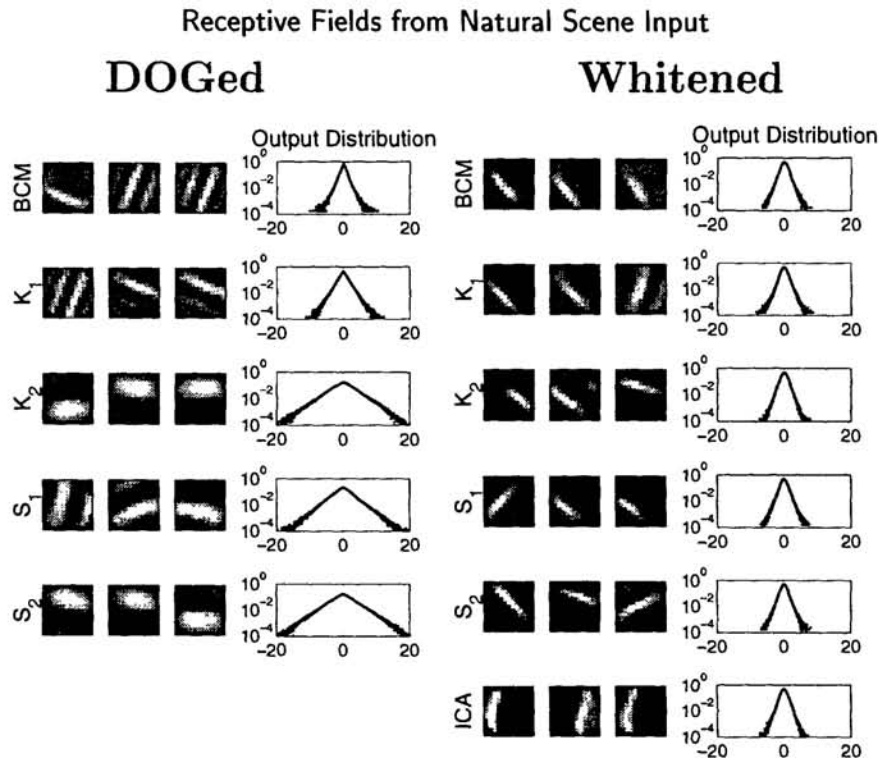

Figure 1: Receptive fields using DOGed (left) and whitened (right) image input obtained from learning rules maximizing (from top to bottom) the Quadratic BCM objective function, Kurtosis (multiplicative), Kurtosis (additive), Skewness (multiplicative), and Skewness (additive). Shown are three examples (left to right) from each learning rule as well as the log of the normalized output distribution, before the application of the rectifying sigmoid.

## 5.2   STRUCTURE REMOVAL: SENSITIVITY TO OUTLIERS

Learning rules which are dependent on large polynomial moments, such as Quadratic BCM and kurtosis, tend to be sensitive to the tails of the distribution. In the case of a sparse code the outliers, or the rare and interesting events, are what is important. Measuring the degree to which the neurons form a sparse code can be done in a straightforward and systematic fashion.

The procedure involves simply eliminating from the environment those patterns for which the neuron responds strongly. These patterns tend to be the high contrast edges, and are thus the structure found in the image. The percentage of patterns that needs to be removed in order to cause a change in the receptive field gives a direct measure of the sparsity of the coding. The results of this structure removal

are shown in Figure 2.

For Quadratic BCM and kurtosis, one need only eliminate *less than one half of a percent* of the input patterns to change the receptive field significantly. To make this more precise, we define a normalized difference between two *mean zero* vectors as $\mathcal{D} \equiv \frac{1}{2}(1 - \cos\alpha)$, where $\alpha$ is the angle between the two vectors. This measure has a value of zero for identical vectors, and a maximum value of one for orthogonal vectors.

Also shown in Figure 2 is the normalized difference as a function of the percentage eliminated, for the different learning rules. RF differences can be seen with as little as a tenth of a percent, which suggests that the neuron is coding the information in a very sparse manner. Changes of around a half a percent and above are visible as significant orientation, phase, or spatial frequency changes. Although both skewness and Quadratic BCM depend primarily on the third moment, QBCM behaves more like kurtosis with regards to sparse coding.

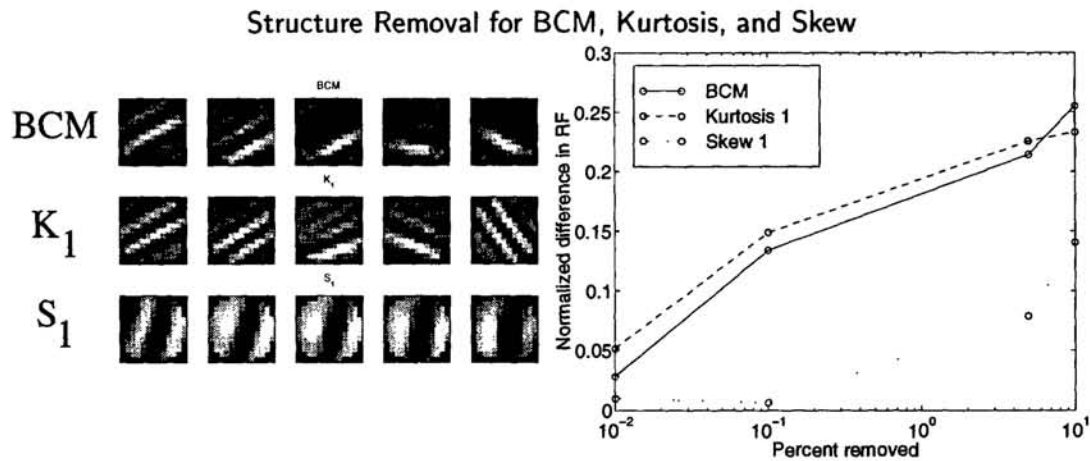

Figure 2: Example receptive fields (left), and normalized difference measure (right), resulting from structure removal using QBCM, $K_1$, and $S_1$. The RFs show the successive deletion of top 1% of the distribution. On the right is the normalized difference between RFs as a function of the percentage deleted in structure removal. The maximum possible value of the difference is 1.

## 6   DISCUSSION

This study attempts to compare several learning rules which have some statistical or biological motivation, or both. For a related study discussing projection pursuit and BCM see (Press and Lee, 1996). We have used natural scenes to gain some more insight about the statistics underlying natural images. There are several outcomes from this study:

- All rules used, found kurtotic distributions.
- The single cell ICA rule we considered, which used the subtractive form of kurtosis, achieved receptive fields qualitatively similar to other rules discussed.
- The Quadratic BCM and the multiplicative version of kurtosis are less sensitive to the second moments of the distribution and produce oriented RFs even when the data is not whitened. The subtractive versions of kurtosis and skewness are sensitive and produces oriented RFs only after sphering the data (Friedman, 1987; Field, 1994).

- Both Quadratic BCM and kurtosis are sensitive to the elimination of the upper 1/2% portion of the distribution. The sensitivity to small portions of the distribution represents the other side of the coin of sparse coding.

- The skew rules' sensitivity to the upper parts of the distribution is not so strong.

- Quadratic BCM learning rule, which has been advocated as a projection index for finding multi-modality in high dimensional distribution, can find projections emphasizing high kurtosis when no cluster structure is present in the data.

## ACKNOWLEDGMENTS

This work, was supported by the Office of Naval Research, the DANA Foundation and the National Science Foundation.

## References

Bell, A. J. and Sejnowski, T. J. (1995). An information-maximisation approach to blind separation and blind deconvolution. *Neural Computation*, 7(6):1129–1159.

Bell, A. J. and Sejnowski, T. J. (1997). The independent components of natural scenes are edge filters. *Vision Research.* in press.

Bienenstock, E. L., Cooper, L. N., and Munro, P. W. (1982). Theory for the development of neuron selectivity: orientation specificity and binocular interaction in visual cortex. *Journal of Neuroscience*, 2:32–48.

Comon, P. (1994). Independent component analysis, a new concept? *Signal Processing*, 36:287–314.

Field, D. J. (1994). What is the goal of sensory coding. *Neural Computation*, 6:559–601.

Friedman, J. H. (1987). Exploratory projection pursuit. *Journal of the American Statistical Association*, 82:249–266.

Hyvarinen, A. and Oja, E. (1996). A fast fixed-point algorithm for independent component analysis. *Int. Journal of Neural Systems*, 7(6):671–687.

Intrator, N. (1990). A neural network for feature extraction. In Touretzky, D. S. and Lippmann, R. P., editors, *Advances in Neural Information Processing Systems*, volume 2, pages 719–726. Morgan Kaufmann, San Mateo, CA.

Intrator, N. and Cooper, L. N. (1992). Objective function formulation of the BCM theory of visual cortical plasticity: Statistical connections, stability conditions. *Neural Networks*, 5:3–17.

Kendall, M. and Stuart, A. (1977). *The Advanced Theory of Statistics*, volume 1. MacMillan Publishing, New York.

Law, C. and Cooper, L. (1994). Formation of receptive fields according to the BCM theory in realistic visual environments. *Proceedings National Academy of Sciences*, 91:7797–7801.

Oja, E. (1995). The nonlinear pca learning rule and signal separation - mathematical analysis. Technical Report A26, Helsinki University, CS and Inf. Sci. Lab.

Olshausen, B. A. and Field, D. J. (1996). Emergence of simple cell receptive field properties by learning a sparse code for natural images. *Nature*, 381:607–609.

Press, W. and Lee, C. W. (1996). Searching for optimal visual codes: Projection pursuit analysis of the statistical structure in natural scenes. In *The Neurobiology of Computation: Proceedings of the fifth annual Computation and Neural Systems conference.* Plenum Publishing Corporation.